# Bayesian Modeling of Facial Similarity

**Baback Moghaddam**
Mitsubishi Electric Research Laboratory
201 Broadway
Cambridge, MA 02139, USA
baback@merl.com

**Tony Jebara and Alex Pentland**
Massachusettes Institute of Technology
20 Ames St.
Cambridge, MA 02139, USA
{jebara,sandy}@media.mit.edu

## Abstract

In previous work [6, 9, 10], we advanced a new technique for direct visual matching of images for the purposes of face recognition and image retrieval, using a *probabilistic* measure of similarity based primarily on a Bayesian (MAP) analysis of image differences, leading to a "dual" basis similar to eigenfaces [13]. The performance advantage of this probabilistic matching technique over standard Euclidean nearest-neighbor eigenface matching was recently demonstrated using results from DARPA's 1996 "FERET" face recognition competition, in which this probabilistic matching algorithm was found to be the top performer. We have further developed a simple method of replacing the costly compution of *nonlinear* (online) Bayesian similarity measures by the relatively inexpensive computation of *linear* (offline) subspace projections and simple (online) Euclidean norms, thus resulting in a significant computational speed-up for implementation with very large image databases as typically encountered in real-world applications.

## 1 Introduction

Current approaches to image matching for visual object recognition and image database retrieval often make use of simple image similarity metrics such as Euclidean distance or normalized correlation, which correspond to a template-matching approach to recognition [2, 5]. For example, in its simplest form, the

similarity measure $S(I_1, I_2)$ between two images $I_1$ and $I_2$ can be set to be inversely proportional to the norm $||I_1 - I_2||$. Such a simple formulation suffers from a major drawback: it does not exploit knowledge of which types of variation are critical (as opposed to incidental) in expressing similarity. In this paper, we formulate a *probabilistic* similarity measure which is based on the probability that the image intensity differences, denoted by $\Delta = I_1 - I_2$, are characteristic of typical variations in appearance of the *same* object. For example, for purposes of face recognition, we can define two classes of facial image variations: *intrapersonal* variations $\Omega_I$ (corresponding, for example, to different facial expressions of the *same* individual) and *extrapersonal* variations $\Omega_E$ (corresponding to variations between *different* individuals). Our similarity measure is then expressed in terms of the probability

$$S(I_1, I_2) \; = \; P(\Delta \in \Omega_I) \; = \; P(\Omega_I | \Delta) \tag{1}$$

where $P(\Omega_I | \Delta)$ is the *a posteriori* probability given by Bayes rule, using estimates of the likelihoods $P(\Delta | \Omega_I)$ and $P(\Delta | \Omega_E)$. The likelihoods are derived from training data using an efficient subspace method for density estimation of high-dimensional data [7, 8]. This Bayesian (MAP) approach can also be viewed as a generalized nonlinear extension of Linear Discriminant Analysis (LDA) [12, 3] or "FisherFace" techniques [1] for face recognition. Moreover, our nonlinear generalization has distinct computational/storage advantages over some of these linear methods for large databases.

## 2  Difference Density Modeling

Consider the problem of characterizing the type of intensity differences which occur when matching two images in a face recognition task. We have two classes (intrapersonal $\Omega_I$ and extrapersonal $\Omega_E$) which we will assume form Gaussian distributions whose likelihoods can be estimated as $P(\Delta | \Omega_I)$ and $P(\Delta | \Omega_E)$ for a given intensity difference $\Delta = I_1 - I_2$.

Given these likelihoods we can evaluate a similarity score $S(I_1, I_2)$ between a pair of images directly in terms of the intrapersonal *a posteriori* probability as given by Bayes rule:

$$S \; = \; \frac{P(\Delta | \Omega_I) P(\Omega_I)}{P(\Delta | \Omega_I) P(\Omega_I) + P(\Delta | \Omega_E) P(\Omega_E)} \tag{2}$$

where the priors $P(\Omega)$ can be set to reflect specific operating conditions (*e.g.*, number of test images *vs.* the size of the database) or other sources of *a priori* knowledge regarding the two images being matched. Additionally, this particular Bayesian formulation casts the standard face recognition task (essentially an $M$-ary classification problem for $M$ individuals) into a *binary* pattern classification problem with $\Omega_I$ and $\Omega_E$. This much simpler problem is then solved using the maximum *a posteriori* (MAP) rule — *i.e.*, two images are determined to belong to the same individual if $P(\Omega_I | \Delta) > P(\Omega_E | \Delta)$, or equivalently, if $S(I_1, I_2) > \frac{1}{2}$.

To deal with the high-dimensionality of $\Delta$, we make use of the efficient density estimation method proposed by Moghaddam & Pentland [7, 8] which divides the vector space $\mathcal{R}^N$ into two complementary subspaces using an eigenspace decomposition. This method relies on a Principal Components Analysis (PCA) [4] to form a low-dimensional estimate of the complete likelihood which can be evaluated using only the first $M$ principal components, where $M \ll N$.

## 3   Efficient Similarity Computation

Consider now a feature space of $\Delta$ vectors, the differences between two images $(I_j$ and $I_k)$. The two classes of interest in this space correspond to intrapersonal and extrapersonal variations and each is modeled as a high-dimensional Gaussian density as in Equation 3. The densities are zero-mean since for each $\Delta = I_j - I_k$ there exists a $\Delta = I_k - I_j$.

$$P(\Delta|\Omega_E) = \frac{e^{-\frac{1}{2}\Delta^T \Sigma_E^{-1}\Delta}}{(2\pi)^{D/2}|\Sigma_E|^{1/2}}$$

$$P(\Delta|\Omega_I) = \frac{e^{-\frac{1}{2}\Delta^T \Sigma_I^{-1}\Delta}}{(2\pi)^{D/2}|\Sigma_I|^{1/2}} \tag{3}$$

By PCA, the Gaussians are known to only occupy a subspace of image space (face-space) and thus, only the top few eigenvectors of the Gaussian densities are relevant for modeling. These densities are used to evaluate the *similarity score* in Equation 2.

Computing the similarity score involves first subtracting a candidate image $I_j$ from a database entry $I_k$. The resulting $\Delta$ image is then projected onto the eigenvectors of the extrapersonal Gaussian and also the eigenvectors of the intrapersonal Gaussian. The exponentials are computed, normalized and then combined as in Equation 2. This operation is iterated over all members of the database (many $I_k$ images) until the maximum score is found (i.e. the match). Thus, for large databases, this evaluation is expensive but can be simplified by offline transformations.

To compute the likelihoods $P(\Delta|\Omega_I)$ and $P(\Delta|\Omega_E)$ we pre-process the $I_k$ images with whitening transformations. Each image is converted and stored as whitened subspace coefficients; **i** for intrapersonal space and **e** for extrapersonal space (see Equation 4). Here, $\Lambda$ and $V$ are matrices of the largest eigenvalues and eigenvectors of $\Sigma_E$ or $\Sigma_I$. Typically, we have used $M_I = 100$ and $M_E = 100$ for $\Omega_I$ and $\Omega_E$ respectively.

$$\mathbf{i}_j = \Lambda_I^{-\frac{1}{2}} V_I I_j \qquad \mathbf{e}_j = \Lambda_E^{-\frac{1}{2}} V_E I_j \tag{4}$$

After this pre-processing, evaluating the Gaussians can be reduced to simple Euclidean distances as in Equation 5. Denominators are of course pre-computed. These likelihoods are evaluated and used to compute the MAP similarity $S$ in Equation 2. Euclidean distances are computed between the 100-dimensional **i** vectors as well as the 100-dimensional **e** vectors. Thus, roughly $2 \times (M_E + M_I) = 400$ arithmetic operations are required for each similarity computation, avoiding repeated image differencing and projections.

$$P(\Delta|\Omega_E) = P(I_j - I_k|\Omega_E) = \frac{e^{-\frac{1}{2}\|\mathbf{e}_j - \mathbf{e}_k\|^2}}{(2\pi)^{D/2}|\Sigma_E|^{1/2}}$$

$$P(\Delta|\Omega_I) = P(I_j - I_k|\Omega_I) = \frac{e^{-\frac{1}{2}\|\mathbf{i}_j - \mathbf{i}_k\|^2}}{(2\pi)^{D/2}|\Sigma_I|^{1/2}} \tag{5}$$

The ML similarity matching is even simpler since only the intra-personal class is evaluated, leading to the following modified form for the similarity measure

$$S' = P(\Delta|\Omega_I) = \frac{e^{-\frac{1}{2}\|\mathbf{i}_j - \mathbf{i}_k\|^2}}{(2\pi)^{D/2}|\Sigma_I|^{1/2}} \tag{6}$$

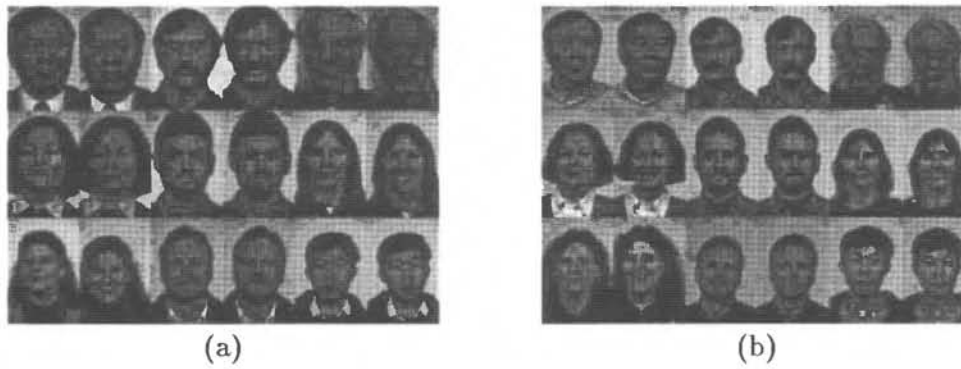

Figure 1: Examples of FERET frontal-view image pairs used for (a) the Gallery set (training) and (b) the Probe set (testing).

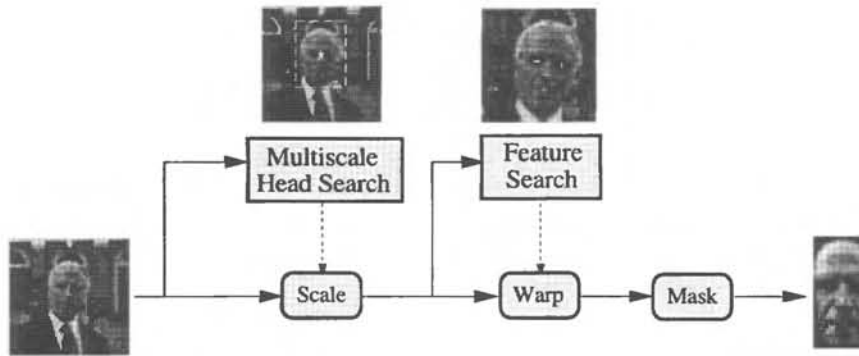

Figure 2: Face alignment system [7].

## 4  Experimental Results

To test our recognition strategy we used a collection of images from the ARPA FERET face database. The set of images consists of pairs of frontal-views (FA/FB) and are divided into two subsets: the "gallery" (training set) and the "probes" (testing set). The gallery images consisted of 74 pairs of images (2 per individual) and the probe set consisted of 38 pairs of images, corresponding to a subset of the gallery members. The probe and gallery datasets were captured a week apart and exhibit differences in clothing, hair and lighting (see Figure 1).

Each of these images were affine normalized with a canonical model using an automatic face-processing system which normalizes for translation, scale as well as slight rotations (both in-plane and out-of-plane). This system is described in detail in [7, 8] and uses maximum-likelihood estimation of object location (in this case the position and scale of a face and the location of individual facial features) to geometrically align faces into standard normalized form as shown in Figure 2. All the faces in our experiments were geometrically aligned and normalized in this manner prior to further analysis.

### 4.1  Eigenface Matching

As a baseline comparison, we first used an eigenface matching technique for recognition [13]. The normalized images from the gallery and the probe sets were projected onto a 100-dimensional eigenspace similar to that shown in Figure 3 and a nearest-neighbor rule based on a Euclidean distance measure was used to match

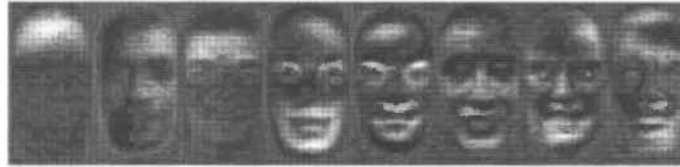

Figure 3: Standard Eigenfaces.

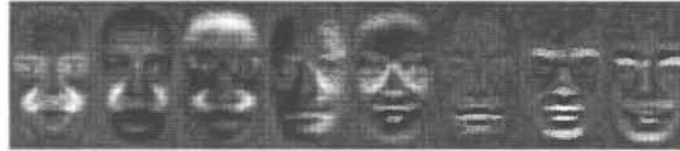

(a)

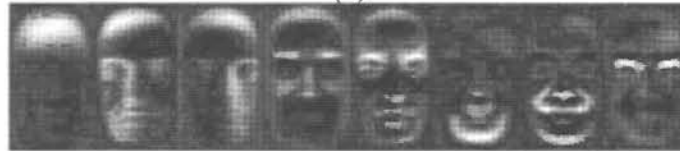

(b)

Figure 4: "Dual" Eigenfaces: (a) Intrapersonal, (b) Extrapersonal

each probe image to a gallery image. We note that this method corresponds to a generalized template-matching method which uses a Euclidean norm measure of similarity which is, however, restricted to the principal subspace of the data. The rank-1 recognition rate obtained with this method was found to be 84%.

## 4.2 Bayesian Matching

For our probabilistic algorithm, we first gathered training data by computing the intensity differences for a training subset of 74 intrapersonal differences (by matching the two views of every individual in the gallery) and a random subset of 296 extrapersonal differences (by matching images of *different* individuals in the gallery), corresponding to the classes $\Omega_I$ and $\Omega_E$, respectively, and performing a separate PCA analysis on each.

We note that the two mutually exclusive classes $\Omega_I$ and $\Omega_E$ correspond to a "dual" set of eigenfaces as shown in Figure 4. Note that the intrapersonal variations shown in Figure 4-(a) represent subtle variations due mostly to expression changes (and lighting) whereas the extrapersonal variations in Figure 4-(b) are more representative of general eigenfaces which code variations such as hair color, facial hair and glasses. These extrapersonal eigenfaces are qualitatively similar to the standard normalized intensity eigenfaces shown in Figure 3.

We next computed the likelihood estimates $P(\Delta|\Omega_I)$ and $P(\Delta|\Omega_E)$ using the PCA-based method [7, 8], using subspace dimensions of $M_I = 10$ and $M_E = 30$ for $\Omega_I$ and $\Omega_E$, respectively. These density estimates were then used with a default setting of equal priors, $P(\Omega_I) = P(\Omega_E)$, to evaluate the *a posteriori* intrapersonal probability $P(\Omega_I|\Delta)$ for matching probe images to those in the gallery. Therefore, for each probe image we computed probe-to-gallery differences and sorted the matching order, this time using the *a posteriori* probability $P(\Omega_I|\Delta)$ as the similarity measure. This probabilistic ranking yielded an improved rank-1 recognition rate of 90%.

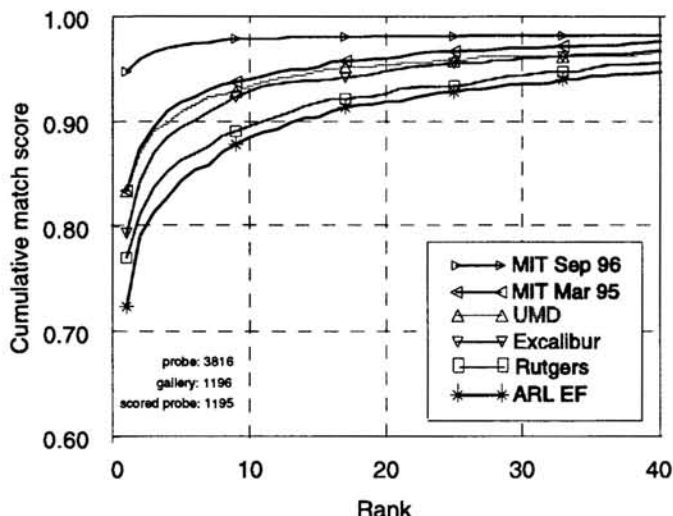

Figure 5: Cumulative recognition rates for frontal FA/FB views for the competing algorithms in the FERET 1996 test. The top curve (labeled "MIT Sep 96") corresponds to our Bayesian matching technique. Note that second placed is standard eigenface matching (labeled "MIT Mar 95").

## 4.3 The 1996 FERET Competition

Our Bayesian approach to recognition has yielded even more significant improvement over simple eigenface techniques with very large face databases. The probabilistic similarity measure was tested in the September 1996 ARPA FERET face recognition competition and yielded a surprising 95% recognition accuracy (on nearly 1200 individuals) making it the top-performing system by a typical margin of 10-20% over the other competing algorithms [11] (see Figure 5). A comparison between standard eigenfaces and the Bayesian method from this test shows a 10% gain in performance afforded by the new similarity measure. Thus we note that, in this particular case, the probabilistic similarity measure has effectively *halved* the error rate of eigenface matching.

Note that we can also use the simplified similarity measure based on the *intrapersonal* eigenfaces for a *maximum likelihood* (ML) matching technique using

$$S' = P(\Delta|\Omega_I) \tag{7}$$

instead of the *maximum a posteriori* (MAP) approach defined by Equation 2. Although this simplified measure has not been officially FERET tested, our own internal experiments with a database of size 2000 have shown that using $S'$ instead of $S$ results in only a minor (2-3%) deficit in the recognition rate while at the same time cutting the computational cost by a further factor of 2.

## 5 Conclusions

The performance advantage of our probabilistic matching technique has been demonstrated using both a small database (internally tested) as well as a large (800+) database with an independent double-blind test as part of ARPA's September 1996 "FERET" competition, in which Bayesian similarity out-performed all competing algorithms (at least one of which was using an LDA/Fisher type method). We believe that these results clearly demonstrate the superior performance of probabilistic matching over eigenface, LDA/Fisher and other existing techniques.

The results obtained with the simplified ML similarity measure ($S'$ in Eq. 7) suggest a computationally equivalent yet superior alternative to standard eigenface matching. In other words, a likelihood similarity based on the intrapersonal density $P(\Delta|\Omega_I)$ alone is far superior to nearest-neighbor matching in eigenspace while essentially requiring the same number of projections. For completeness (and a slightly better performance) however, one should use the *a posteriori* similarity $S$ in Eq. 2, at twice the computational cost of standard eigenfaces.

This probabilistic framework is particularly advantageous in that the intra/extra density estimates explicitly characterize the type of appearance variations which are critical in formulating a meaningful measure of similarity. For example, the deformations corresponding to facial expression changes (which may have high image-difference norms) are, in fact, *irrelevant* when the measure of similarity is to be based on *identity*. The subspace density estimation method used for representing these classes thus corresponds to a *learning* method for discovering the principal modes of variation important to the classification task.

# References

[1] V.I. Belhumeur, J.P. Hespanha, and D.J. Kriegman. Eigenfaces vs. fisherfaces: Recognition using class specific linear projection. *IEEE Transactions on Pattern Analysis and Machine Intelligence*, PAMI-19(7):711–720, July 1997.

[2] R. Brunelli and T. Poggio. Face recognition : Features vs. templates. *IEEE Transactions on Pattern Analysis and Machine Intelligence*, 15(10), October 1993.

[3] K. Etemad and R. Chellappa. Discriminant analysis for recognition of human faces. In *Proc. of Int'l Conf. on Acoustics, Speech and Signal Processing*, pages 2148–2151, 1996.

[4] I.T. Jolliffe. *Principal Component Analysis*. Springer-Verlag, New York, 1986.

[5] M. J. Jones and T. Poggio. Model-based matching by linear combination of prototypes. AI Memo No. 1583, Artificial Intelligence Laboratory, Massachusettes Institute of Technology, November 1996.

[6] B. Moghaddam, C. Nastar, and A. Pentland. Bayesian face recognition using deformable intensity differences. In *Proc. of IEEE Conf. on Computer Vision and Pattern Recognition*, June 1996.

[7] B. Moghaddam and A. Pentland. Probabilistic visual learning for object detection. In *IEEE Proceedings of the Fifth International Conference on Computer Vision (ICCV'95)*, Cambridge, USA, June 1995.

[8] B. Moghaddam and A. Pentland. Probabilistic visual learning for object representation. *IEEE Transactions on Pattern Analysis and Machine Intelligence*, PAMI-19(7):696–710, July 1997.

[9] B. Moghaddam, W. Wahid, and Alex Pentland. Beyond eigenfaces: Probabilistic matching for face recognition. In *Proc. of Int'l Conf. on Automatic Face and Gesture Recognition*, pages 30–35, Nara, Japan, April 1998.

[10] C. Nastar, B. Moghaddam, and A. Pentland. Generalized image matching: Statistical learning of physically-based deformations. In *Proceedings of the Fourth European Conference on Computer Vision (ECCV'96)*, Cambridge, UK, April 1996.

[11] P. J. Phillips, H. Moon, P. Rauss, and S. Rizvi. The FERET evaluation methodology for face-recognition algorithms. In *IEEE Proceedings of Computer Vision and Pattern Recognition*, pages 137–143, June 1997.

[12] D. Swets and J. Weng. Using discriminant eigenfeatures for image retrieval. *IEEE Transactions on Pattern Analysis and Machine Intelligence*, PAMI-18(8):831–836, August 1996.

[13] M. Turk and A. Pentland. Eigenfaces for recognition. *Journal of Cognitive Neuroscience*, 3(1), 1991.
